# Complexity Issues in Neural Computation and Learning

**V. P. Roychowdhury**
School of Electrical Engineering
Purdue University
West Lafayette, IN 47907
Email: vwani@ecn.purdue.edu

**K.-Y. Siu**
Dept. of Electrical & Comp. Engr.
University of California at Irvine
Irvine, CA 92717
Email: siu@balboa.eng.uci.edu

The general goal of this workshop was to bring together researchers working toward developing a theoretical framework for the analysis and design of neural networks. The technical focus of the workshop was to address recent developments in understanding the capabilities and limitations of various models for neural computation and learning. The primary topics addressed the following three areas: 1) Computational complexity issues in neural networks, 2) Complexity issues in learning, and 3) Convergence and numerical properties of learning algorithms. Other topics included experimental/simulation results on neural networks, which seemed to pose some open problems in the areas of learning and generalization properties of feedforward networks.

The presentations and discussions at the workshop highlighted the interdisciplinary nature of research in neural networks. For example, several of the presentations discussed recent contributions which have applied complexity-theoretic techniques to characterize the computing power of neural networks, to design efficient neural networks, and to compare the computational capabilities of neural networks with that of conventional models for computation. Such studies, in turn, have generated considerable research interest among computer scientists, as evidenced by a significant number of research publications on related topics. A similar development can be observed in the area of learning as well: Techniques primarily developed in the classical theory of learning are being applied to understand the generalization and learning characteristics of neural networks. In [1, 2] attempts have been made to integrate concepts from different areas and present a unified treatment of the various results on the complexity of neural computation and learning. In fact, contributions from several participants in the workshop are included in [2], and interested readers could find detailed discussions of many of the results presented at the workshop in [2].

Following is a brief description of the presentations, along with the names and e-mail addresses of the speakers. W. Maass (*maass@igi.tu-graz.ac.at*) and A. Sakurai (*sakurai@harlgw92.harl.hitachi.co.jp*) made presentations on the VC-dimension and the computational power of feedforward neural networks. Many neural nets of depth 3 (or larger) with linear threshold gates have a VC-dimension that is superlinear in the number of weights of the net. The talks presented new results which establish

effective upper bounds and almost tight lower bounds on the VC-dimension of feedforward networks with various activation functions including linear threshold and sigmoidal functions. Such nonlinear lower bounds on the VC-dimension were also discussed for networks with both integer and real weights. A presentation by G. Turán (*@VM.CC.PURDUE.EDU:U11557@UICVM*) discussed new results on proving lower bounds on the size of circuits for computing specific Boolean functions where each gate computes a real-valued function. In particular the results provide a lower bound for the size of formulas (i.e., circuits with fan-out 1) of polynomial gates, computing Boolean functions in the sense of sign-representation.

The presentations on learning addressed both sample and algorithmic complexity. The talk by V. Castelli (*vittorio@isl.stanford.edu*) and T. Cover studied the role of labeled and unlabeled samples in pattern recognition. Let samples be chosen from two populations whose distributions are known, and let the proportion (mixing parameter) of the two classes be unknown. Assume that a training set composed of independent observations from the two classes is given, where part of the samples are classified and part are not. The talk presented new results which investigate the relative value of the labeled and unlabeled samples in reducing the probability of error of the classifier. In particular, it was shown that under the above hypotheses the relative value of labeled and unlabeled samples is proportional to the (Fisher) Information they carry about the unknown mixing parameter. B. Dasgupta (*dasgupta@cs.umn.edu*), on the other hand, addressed the issue of the tractability of the training problem of neural networks. New results showing that the training problem remains NP-complete when the activation functions are piecewise linear were presented.

The talk by B. Hassibi (*hassibi@rascals.stanford.edu*) provided a minimax interpretation of instantaneous-gradient-based learning algorithms such as LMS and backpropagation. When the underlying model is linear, it was shown that the LMS algorithm minimizes the worst case ratio of predicted error energy to disturbance energy. When the model is nonlinear, which arises in the context of neural networks, it was shown that the backpropagation algorithm performs this minimization in a *local* sense. These results provide theoretical justification for the widely observed excellent robustness properties of the LMS and backpropagation algorithms.

The last talk by R. Caruana (*caruana@GS79.SP.CS.CMU.EDU*) presented a set of interesting empirical results on the learning properties of neural networks of different sizes. Some of the issues (based on empirical evidence) raised during the talk are: 1) If cross-validation is used to prevent overtraining, excess capacity rarely reduces the generalization performance of fully connected feed-forward backpropagation networks. 2) Moreover, too little capacity usually hurts generalization performance more than too much capacity.

# References

[1] K.-Y. Siu, V. P. Roychowdhury, and T. Kailath. *Discrete Neural Computation: A Theoretical Foundation*. Englewood Cliffs, NJ: Prentice-Hall, 1994.

[2] V. P. Roychowdhury, K.-Y. Siu, and A. Orlitsky, editors. *Theoretical Advances in Neural Computation and Learning*. Boston: Kluwer Academic Publishers, 1994.
